# Means, Correlations and Bounds

**M.A.R. Leisink and H.J. Kappen**
Department of Biophysics
University of Nijmegen, Geert Grooteplein 21
NL 6525 EZ Nijmegen, The Netherlands
{martijn,bert}@mbfys.kun.nl

## Abstract

The partition function for a Boltzmann machine can be bounded from above and below. We can use this to bound the means and the correlations. For networks with small weights, the values of these statistics can be restricted to non-trivial regions (i.e. a subset of $[-1, 1]$). Experimental results show that reasonable bounding occurs for weight sizes where mean field expansions generally give good results.

## 1 Introduction

Over the last decade, bounding techniques have become a popular tool to deal with graphical models that are too complex for exact computation. A nice property of bounds is that they give at least some information you can rely on. For instance, one may find that a correlation is definitely between 0.4 and 0.6. An ordinary approximation might be more accurate, but in practical situations there is absolutely no warranty for that.

The best known bound is probably the mean field bound, which has been described for Boltzmann machines in [1] and later for sigmoid belief networks in [2]. Apart from its bounding properties, mean field theory is a commonly used approximation technique as well. Recently this first order bound was extended to a third order approximation for Boltzmann machines and sigmoid belief networks in [3] and [4], where it was shown that this particular third order expansion is still a bound.

In 1996 an upper bound for Boltzmann machines was described in [5] and [6]. In the same articles the authors derive an upper bound for a special case of sigmoid belief networks: the two-layered networks. In this article we will focus solely on Boltzmann machines, but an extension to sigmoid belief networks is quite straightforward.

This article is organized as follows: In section 2 we start with the general theory about bounding techniques. Later in that section the upper and lower bound are briefly described. For a full explanation we refer to the articles mentioned before. The section is concluded by explaining how these bounds on the partition function can be used to bound means and correlations. In section 3 results are shown for fully connected Boltzmann machines, where size of weights and thresholds as well as network size are varied. In section 4 we present our conclusions and outline possible extensions.

## 2 Theory

There exists a general method to create a class of polynomials of a certain order, which all bound a function of interest, $f_0(x)$. Such a class of order $2n$ can be found if the $2n$-th order derivative of $f_0(x)$, written as $f_{2n}(x)$, can be bounded by a constant. When this constant is zero, the class is actually of order $2n-1$. It turns out that this class is parameterized by $n$ free parameters.

Suppose we have a function $b_{2k}$ for some integer $k$ which bounds the function $f_{2k}$ from below (an upper bound can be written as a lower bound by using the negative of both functions). Thus

$$\forall_x \quad f_{2k}(x) \geq b_{2k}(x) \tag{1}$$

Now construct the primitive functions $f_{2k-1}$ and $b_{2k-1}$ such that $f_{2k-1}(\mu) = b_{2k-1}(\mu)$ for a free to choose value for $\mu$. This constraint can always be achieved by adding an appropriate constant to the primitive function $b_{2k-1}$. It is easy to prove that

$$\begin{cases} f_{2k-1}(x) \leq b_{2k-1}(x) & \text{for } x \leq \mu \\ f_{2k-1}(x) \geq b_{2k-1}(x) & \text{for } x \geq \mu \end{cases} \tag{2}$$

or in shorthand notation $f_{2k-1}(x) \lesseqgtr b_{2k-1}(x)$.

If we repeat this procedure and construct the primitive functions $f_{2k-2}$ and $b_{2k-2}$ such that $f_{2k-2}(\mu) = b_{2k-2}(\mu)$ for the same $\mu$, one can verify that

$$\forall_x \quad f_{2k-2}(x) \geq b_{2k-2}(x) \tag{3}$$

Thus given a bound $f_{2k}(x) \geq b_{2k}(x)$ we can construct a class of bounding functions for $f_{2k-2}$ parameterized by $\mu$.

Since we assumed $f_{2n}(x)$ can be bounded from below by a constant, we can apply the procedure $n$ times and we finally find $f_0(x) \geq b_0(x)$, where $b_0(x)$ is parameterized by $n$ free parameters. This procedure can be found in more detail in [4].

### 2.1 A third order lower bound for Boltzmann machines

Boltzmann machines are stochastic neural networks with $N$ binary valued neurons, $s_i$, which are connected by symmetric weights $w_{ij}$. Due to this symmetry the probability distribution is a Boltzmann-Gibbs distribution which is given by (see also [7])

$$p(\vec{s}|\theta, w) = \frac{1}{Z} \exp\left(\frac{1}{2}\sum_{ij} w_{ij} s_i s_j + \sum_i \theta_i s_i\right) = \frac{1}{Z} \exp\left(-E(\vec{s}, \theta, w)\right) \tag{4}$$

where the $\theta_i$ are threshold values and

$$Z(\theta, w) = \sum_{\text{all } \vec{s}} \exp\left(-E(\vec{s}, \theta, w)\right) \tag{5}$$

is the normalization known as the partition function.

This partition function is especially important, since statistical quantities such as means and correlations can be directly derived from it. For instance, the means can be computed as

$$\langle s_n \rangle = \sum_{\text{all } \vec{s}} p(\vec{s}|\theta, w) s_n = \sum_{\text{all } \vec{s}/s_n} p(\vec{s}, s_n = +1|\theta, w) - p(\vec{s}, s_n = -1|\theta, w)$$

$$= \frac{Z_+(\theta, w) - Z_-(\theta, w)}{Z(\theta, w)} \tag{6}$$

where $Z_+$ and $Z_-$ are partition functions over a network with $s_n$ clamped to $+1$ and $-1$, respectively.

This explains why the objective of almost any approximation method is the partition function given by equation 5. In [3] and [4] it is shown that the standard mean field lower bound can be obtained by applying the linear bound

$$\forall_{x,\mu} \quad e^x \geq e^\mu \left(1 + x - \mu\right) \tag{7}$$

to all exponentially many terms in the sum. Since $\mu$ may depend on $\vec{s}$, one can choose $\mu\left(\vec{s}\right) = \mu_i s_i + \mu_0$, which leads to the standard mean field equations, where the $\mu_i$ turn out to be the local fields.

Moreover, the authors show that one can apply the procedure of 'upgrading bounds' (which is described briefly at the beginning of this section) to equation 7, which leads to the class of third order bounds for $e^x$. This is achieved in the following way:

$$\forall_{x,\nu} \; f_2(x) = e^x \geq e^\nu \left(1 + x - \nu\right) = b_2(x)$$

$$f_1(x) = e^x \lessgtr e^\mu + e^\nu \left((1 + \mu - \nu)(x - \mu) + \frac{1}{2}(x-\mu)^2\right) = b_1(x) \tag{8}$$

$$\forall_{x,\mu,\lambda} \; f_0(x) = e^x \geq e^\mu \left\{1 + x - \mu + e^\lambda \left(\frac{1-\lambda}{2}(x-\mu)^2 + \frac{1}{6}(x-\mu)^3\right)\right\} = b_0(x)$$

with $\lambda = \nu - \mu$.

In principle, this third order bound could be maximized with respect to all the free parameters, but here we follow the suggestion made in [4] to use a mean field optimization, which is much faster and generally almost as good as a full optimization. For more details we refer to [4].

## 2.2 An upper bound

An upper bound for Boltzmann machines has been described in [5] and [6][1]. Basically, this method uses a quadratic upper bound on $\log \cosh x$, which can easily be obtained in the following way:

$$f_2(x) = 1 - \tanh^2 x \leq 1 = b_2(x)$$

$$f_1(x) = \tanh x \gtrless x - \mu + \tanh \mu = b_1(x) \tag{9}$$

$$f_0(x) = \log \cosh x \leq \frac{1}{2}(x-\mu)^2 + (x-\mu)\tanh \mu + \log \cosh \mu = b_0(x)$$

Using this bound, one can derive

$$Z\left(\theta, w\right) = \sum_{\text{all } \vec{s}} \exp\left(\frac{1}{2}\sum_{ij} w_{ij}s_i s_j + \sum_i \theta_i s_i\right)$$

$$= \sum_{\text{all } \vec{s}/s_n} 2\exp\left(\log \cosh\left(\sum_i w_{ni}s_i + \theta_n\right)\right)\exp\left(\frac{1}{2}\sum_{ij \neq n} w_{ij}s_i s_j + \sum_{i \neq n}\theta_i s_i\right)$$

$$\leq \sum_{\text{all } \vec{s}/s_n} \exp\left(\frac{1}{2}\sum_{ij \neq n} w'_{ij}s_i s_j + \sum_{i \neq n}\theta'_i s_i + k\right) = e^k \cdot Z\left(\theta', w'\right) \tag{10}$$

where $k$ is a constant and $\theta'$ and $w'$ are thresholds and weights in a reduced network given by

$$w'_{ij} = w_{ij} + w_{ni} w_{nj}$$
$$\theta'_{ij} = \theta_i + w_{ni} \left( \theta_n - \mu_n + \tanh \mu_n \right) \tag{11}$$
$$k = \frac{1}{2} \left( \theta_n - \mu_n + \tanh \mu_n \right)^2 - \frac{1}{2} \tanh^2 \mu_n + \log 2 \cosh \mu_n$$

Hence, equation 10 defines a recursive relation, where each step reduces the network by one neuron. Finally, after $N$ steps, an upper bound on the partition function is found[2]. We did a crude minimization with respect to the free parameters $\mu$. A more sophisticated method can probably be found, but this is not the main objective of this article.

### 2.3 Bounding means and correlations

The previous subsections showed very briefly how we can obtain a lower bound, $Z^{\mathrm{L}}$, and an upper bound, $Z^{\mathrm{U}}$, for any partition function. We can use this in combination with equation 6 to obtain a bound on the means:

$$\langle s_n \rangle^{\mathrm{L}} = \frac{Z^{\mathrm{L}}_+ - Z^{\mathrm{U}}_-}{X} \le \langle s_n \rangle \le \frac{Z^{\mathrm{U}}_+ - Z^{\mathrm{L}}_-}{Y} = \langle s_n \rangle^{\mathrm{U}} \tag{12}$$

where $X = Z^{\mathrm{U}}$ if the nominator is positive and $X = Z^{\mathrm{L}}$ otherwise. For $Y$ it is the opposite. The difference, $\langle s_n \rangle^{\mathrm{U}} - \langle s_n \rangle^{\mathrm{L}}$, is called the bandwidth.

Naively, we can compute the correlations similarly to the means using

$$\langle s_n s_m \rangle = \frac{Z_{++} + Z_{--} - Z_{+-} - Z_{-+}}{Z} \tag{13}$$

where the partition function is computed for all combinations $s_n s_m$. Generally, however, this gives poor results, since we have to add four bounds together, which leads to a bandwidth which is about twice as large as for the means. We can circumvent this by computing the correlations using

$$\langle s_n s_m \rangle = \frac{Z \left( \theta, w | s_n = s_m \right) - Z \left( \theta, w | s_n = -s_m \right)}{Z} \tag{14}$$

where we allow the sum in the partition functions to be taken over $s_n$, but fix $s_m$ either to $s_n$ or its negative. Finally, the computation of the bounds $\langle s_n s_m \rangle^{\mathrm{L}}$ and $\langle s_n s_m \rangle^{\mathrm{U}}$ is analogue to equation 12.

There exists an alternative way to bound the means and correlations. One can write

$$\langle s_n \rangle = \frac{Z_+ - Z_-}{Z_+ + Z_-} = \frac{Z_+/Z_- - 1}{Z_+/Z_- + 1} = \frac{z - 1}{z + 1} = f(z) \tag{15}$$

with $z = Z_+/Z_-$, which can be bounded by

$$\frac{Z^{\mathrm{L}}_+}{Z^{\mathrm{U}}_-} \le z \le \frac{Z^{\mathrm{U}}_+}{Z^{\mathrm{L}}_-} \tag{16}$$

Since $f(z)$ is a monotonically increasing function of $z$, the bounds on $\langle s_n \rangle$ are given by applying this function to the left and right side of equation 16. The correlations can be bounded similarly. It is still unknown whether this algorithm would yield better results than the first one, which is explored in this article.

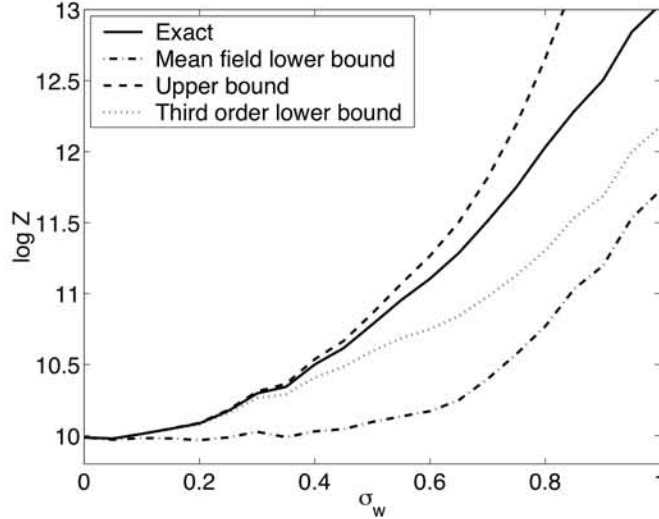

Figure 1: Comparison of 1) the mean field lower bound, 2) the upper bound and 3) the third order lower bound with the exact log partition function. The network was a fully connected Boltzmann machine with 14 neurons and $\sigma_\theta = 0.2$. The size of the weights is varied on the $x$-axis. Each point was averaged over 100 networks.

## 3  Results

In all experiments we used fully connected Boltzmann machines of which the thresholds and weights both were drawn from a Gaussian with zero mean and standard deviation $\sigma_\theta$ and $\sigma_w / \sqrt{N}$, respectively, where $N$ is the network size. This is the so called SK-model (see also [8]). Generally speaking, the mean field approximation breaks down for $\sigma_\theta = 0$ and $\sigma_w > 0.5$, whereas it can be proven that any expansion based approximation is inaccurate when $\sigma_w > 1$ (which is the radius of convergence as in [9]). If $\sigma_\theta \neq 0$ these maximum values are somewhat larger.

In figure 1 we show the logarithm of the exact partition function, the first order or mean field bound, the upper bound (which is roughly quadratic) and the third order lower bound. The weight size is varied along the horizontal axis. One can see clearly that the mean field bound is not able to capture the quadratic form of the exact partition function for small weights due to its linear behaviour. The error made by the upper and third order lower bound is small enough to make non-trivial bounds on the means and correlations.

An example of this bound is shown in figure 2 for the specific choice $\sigma_\theta = \sigma_w = 0.4$. For both the means and the correlations a histogram is plotted for the upper and lower bounds computed with equation 12. Both have an average bandwidth of 0.132, which is a clear subset of the whole possible interval of $[-1, 1]$.

In figure 3 the average bandwidth is shown for several values of $\sigma_\theta$ and $\sigma_w$. For bandwidths of 0.01, 0.1 and 1 a line is drawn. We conclude that almost everywhere the bandwidth is non-trivially reduced and reaches practically useful values for $\sigma_w$ less than 0.5. This is more or less equivalent to the region where the mean field approximation performs well. That approximation, however, gives no information on how close it actually is to the exact value, whereas the bounding method limits it to a definite region.

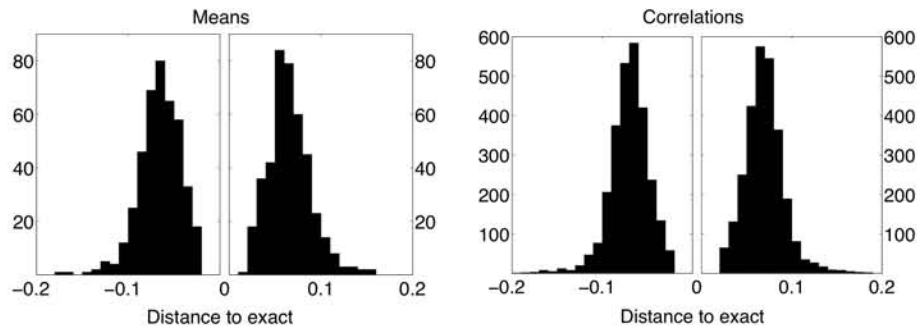

Figure 2: For the specific choice $\sigma_\theta = \sigma_w = 0.4$ thirty fully connected Boltzmann machines with 14 neurons were initialized and the bounds were computed. The two left panels show the distance between the lower bound and the exact means (left) and similarly for the upper bound (right). The right two panels show the distances of both bounds for the correlations.

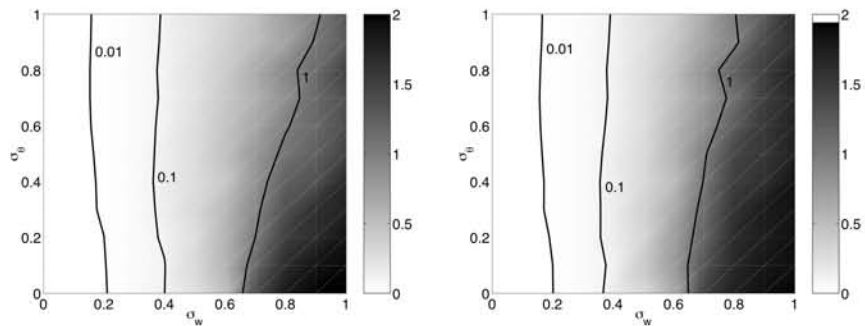

Figure 3: In the left panel the average bandwidth is colour coded for the means, where $\sigma_\theta$ and $\sigma_w$ are varied in ten steps along the axes. The right panel shows the same for the correlations. For each $\sigma_\theta, \sigma_w$ thirty fully connected Boltzmann machines were initialized and the bounds on all the means and correlations were computed. For three specific bandwidths a line is drawn.

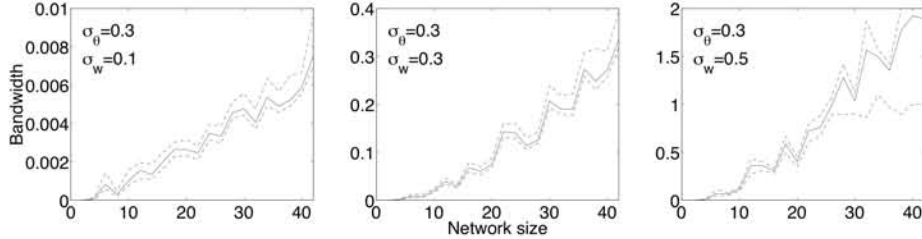

Figure 4: For $\sigma_w = 0.1$, 0.3 and 0.5 the bandwidth for the correlations is shown versus the network size. $\sigma_\theta = 0.3$ in all cases, but the plots are nearly the same for other values. Please note the different scales for the $y$-axis. A similar graph for the means is not shown here, but it is roughly the same. The solid line is the average bandwidth over all correlations, whereas the dashed lines indicate the minimum and maximum bandwidth found.

Unfortunately, the bounds have the unwanted property that the error scales badly with the size of the network. Although this makes the bounds unsuitable for very large networks, there is still a wide range of networks small enough to take advantage of the proposed method and still much too large to be treated exactly. The bandwidth versus network size is shown in figure 4 for three values of $\sigma_w$. Obviously, the threshold of practical usefulness is reached earlier for larger weights.

Finally, we remark that the computation time for the upper bound is $\mathcal{O}\left(N^4\right)$ and $\mathcal{O}\left(N^3\right)$ for the mean field and third order lower bound. This is not shown here.

## 4 Conclusions

In this article we combined two already existing bounds in such a way that not only the partition function of a Boltzmann machine is bounded from both sides, but also the means and correlations. This may seem superfluous, since there exist already several powerful approximation methods. Our method, however, can be used apart from any approximation technique and gives at least some information you can rely on. Although approximation techniques might do a good job on your data, you can never be sure about that. The method outlined in this paper ensures that the quantities of interest, the means and correlations, are restricted to a certain region.

We have seen that, generally speaking, the results are useful for weight sizes where an ordinary mean field approximation performs well. This makes the method applicable to a large class of problems. Moreover, since many architectures are not fully connected, one can take advantage of that structure. At least for the upper bound it is shown already that this can improve computation speed and tightness. This would partially cancel the unwanted scaling with the network size.

Finally, we would like to give some directions for further research. First of all, an extension to sigmoid belief networks can easily be done, since both a lower and an upper bound are already described. The upper bound, however, is only applicable to two layer networks. A more general upper bound can probably be found. Secondly one can obtain even better bounds (especially for larger weights) if the general constraint

$$\forall_{nm} \quad -1 + |\langle s_n \rangle + \langle s_m \rangle| \leq \langle s_n s_m \rangle \leq 1 - |\langle s_n \rangle - \langle s_m \rangle| \tag{17}$$

is taken into account. This might even be extended to similar constraints, where three or more neurons are involved.

**Acknowledgements**

This research is supported by the Technology Foundation STW, applied science devision of NWO and the technology programme of the Ministry of Economic Affairs.

## Footnotes

[1]Note: The articles referred to, use $s_i \in \{0, 1\}$ instead of the $+1/-1$ coding used here.

[2]The original articles show that it is not necessary to do all the $N$ steps. However, since this is based on mixing approximation techniques with exact calculations, it is not used here as it would hide the real error the approximation makes.

# References

[1] C. Peterson and J. Anderson. A mean field theory learning algorithm for neural networks. *Complex systems*, 1:995–1019, 1987.

[2] S.K. Saul, T.S. Jaakkola, and M.I. Jordan. Mean field theory for sigmoid belief networks. *Journal of Artificial Intelligence Research*, 4:61–76, 1996.

[3] Martijn A.R. Leisink and Hilbert J. Kappen. A tighter bound for graphical models. In Todd K. Leen, Thomas G. Dietterich, and Volker Tresp, editors, *Advances in Neural Information Processing Systems 13*, pages 266–272. MIT Press, 2001.

[4] Martijn A.R. Leisink and Hilbert J. Kappen. A tighter bound for graphical models. *Neural Computation*, 13(9), 2001. To appear.

[5] T. Jaakkola and M.I. Jordan. Recursive algorithms for approximating probabilities in graphical models. *MIT Comp. Cogn. Science Technical Report 9604*, 1996.

[6] Tommi S. Jaakkola and Michael I. Jordan. Computing upper and lower bounds on likelihoods in intractable networks. In *Proceedings of the Twelfth Annual Conference on Uncertainty in Artificial Intelligence (UAI-96)*, pages 340–348, San Francisco, CA, 1996. Morgan Kaufmann Publishers.

[7] D. Ackley, G. Hinton, and T. Sejnowski. A learning algorithm for Boltzmann machines. *Cognitive Science*, 9:147–169, 1985.

[8] D. Sherrington and S. Kirkpatrick. Solvable model of a spin-glass. *Physical Review Letters*, 35(26):1793–1796, 12 1975.

[9] T. Plefka. Convergence condition of the TAP equation for the infinite-ranged ising spin glass model. *J.Phys.A: Math.Gen.*, 15:1971–1978, 1981.
